# Clustering via Dirichlet Process Mixture Models for Portable Skill Discovery

**Scott Niekum**        **Andrew G. Barto**
Department of Computer Science
University of Massachusetts Amherst
Amherst, MA 01003
{sniekum,barto}@cs.umass.edu

## Abstract

Skill discovery algorithms in reinforcement learning typically identify single states or regions in state space that correspond to task-specific subgoals. However, such methods do not directly address the question of how many distinct skills are appropriate for solving the tasks that the agent faces. This can be highly inefficient when many identified subgoals correspond to the same underlying skill, but are all used individually as skill goals. Furthermore, skills created in this manner are often only transferable to tasks that share identical state spaces, since corresponding subgoals across tasks are not merged into a single skill goal. We show that these problems can be overcome by clustering subgoal data defined in an *agent-space* and using the resulting clusters as templates for skill termination conditions. Clustering via a Dirichlet process mixture model is used to discover a minimal, sufficient collection of portable skills.

## 1   Introduction

Reinforcement learning (RL) is often used to solve single tasks for which it is tractable to learn a good policy with minimal initial knowledge. However, many real-world problems cannot be solved in this fashion, motivating recent research on transfer and hierarchical RL methods that allow knowledge to be generalized to new problems and encapsulated in modular skills. Although skills have been shown to improve agent learning performance [2], representational power [10], and adaptation to non-stationarity [3], to the best of our knowledge, current methods lack the ability to automatically discover skills that are transferable to related state spaces and novel tasks, especially in continuous domains.

Skill discovery algorithms in reinforcement learning typically identify single states or regions in state space that correspond to task-specific subgoals. However, such methods do not directly address the question of how many distinct skills are appropriate for solving the tasks that the agent faces. This can be highly inefficient when many identified subgoals correspond to the same underlying skill, but are all used individually as skill goals. For example, opening a door ought to be the same skill whether an agent is one inch or two inches away from the door, or whether the door is red or blue; making each possible configuration a separate skill would be unwise. Furthermore, skills created in this manner are often only transferable to tasks that share identical state spaces, since corresponding subgoals across tasks are not merged into a single skill goal.

We show that these problems can be overcome by collecting subgoal data from a series of tasks and clustering it in an *agent-space* [9], a shared feature space across multiple tasks. The resulting clusters generalize subgoals within and across tasks and can be used as templates for portable skill termination conditions. Clustering also allows the creation of skill termination conditions in a data-driven way that makes minimal assumptions and can be tailored to the domain through a careful

choice of clustering algorithm. Additionally, this framework extends the utility of single-state sub-goal discovery algorithms to continuous domains, in which the agent may never see the same state twice. We argue that clustering based on a Dirichlet process mixture model is appropriate in the general case when little is known about the nature or number of skills needed in a domain. Experiments in a continuous domain demonstrate the utility of this approach and illustrate how it may be useful even when traditional subgoal discovery methods are infeasible.

## 2 Background and Related Work

### 2.1 Reinforcement learning

The RL paradigm [20] usually models a problem faced by the agent as a Markov decision process (MDP), expressed as $M = \langle S, A, P, R \rangle$, where $S$ is the set of environment states the agent can observe, $A$ is the set of actions that the agent can execute, $P(s, a, s')$ is the probability that the environment transitions to $s' \in S$ when action $a \in A$ is taken in state $s \in S$, and $R(s, a, s')$ is the expected scalar reward given to the agent when the environment transitions to state $s'$ from $s$ after the agent takes action $a$.

### 2.2 Options

The options framework [19] models skills as temporally extended actions that can be invoked like primitive actions. An option $o$ consists of an *option policy* $\pi_o : S \times A \to [0, 1]$, giving the probability of taking action $a$ in state $s$, an *initiation set* $\mathcal{I}_o \subseteq S$, giving the set of states from which the option can be invoked, and a *termination condition* $\beta_o : S \to [0, 1]$, giving the probability that option execution will terminate upon reaching state $s$. In this paper, termination conditions are binary, so that we can define a *termination set* of states, $\mathcal{T}_o \subseteq S$, in which option execution always terminates.

### 2.3 Agent-spaces

To facilitate option transfer across multiple tasks, Konidaris and Barto [9] propose separating problems into two representations. The first is a *problem-space* representation which is Markov for the current task being faced by the agent, but may change across tasks; this is the typical formulation of a problem in RL. The second is an *agent-space* representation, which is identical across all tasks to be faced by the agent, but may not be Markov for any particular task. An agent-space is often a set of agent-centric features, like a robot's sensor readings, that are present and retain semantics across tasks. If the agent represents its top-level policy in a task-specific problem-space but represents its options in an agent-space, the task at hand will always be Markov while allowing the options to transfer between tasks.

Agent-spaces enable the transfer of an option's policy between tasks, but are based on the assumption that this policy was learned under an option termination set that is portable; the termination set must accurately reflect how the goal of the skill varies across tasks. Previous work using agent-spaces has produced portable option policies when the termination sets were hand-coded; our contribution is the automatic discovery of portable termination sets, so that such skills can be aquired autonomously.

### 2.4 Subgoal discovery and skill creation

The simplest subgoal discovery algorithms analyze reward statistics or state visitation frequencies to discover subgoal states [3]. Graph-based algorithms [18, 11] search for "bottleneck" states on state transition graphs via clustering and other types of analysis. Algorithms based on intrinsic motivation have included novelty metrics [17] and hand-coded salience functions [2]. Skill chaining [10] discovers subgoals by 'chaining' together options, in which the termination set of one option is the empirically determined initiation set of the next option in the chain. HASSLE [1] clusters similar regions of state space to identify single-task subgoals. All of these methods compute subgoals that may be inefficient or non-portable if used alone as skill targets, but that can be used as data for our algorithm to find portable options.

Other algorithms analyze tasks to create skills directly, rather than search for subgoals. VISA [7] creates skills to control factored state variables in tasks with sparse causal graphs. PolicyBlocks

[15] looks for policy similarities that can be used as templates for skills. The SKILLS algorithm [21] attempts to minimize description length of policies while preserving a performance metric. However, these methods only exhibit transfer to identical state spaces and often rely on discrete state representations. Related work has also used clustering to determine which of a set of MDPs an agent is currently facing, but does not address the need for skills within a single MDP [22].

## 2.5 Dirichlet process mixture models

Many popular clustering algorithms require the number of data clusters to be known *a priori* or use heuristics to choose an approximate number. By contrast, Dirichlet process mixture models (DP-MMs) provide a non-parametric Bayesian framework to describe distributions over mixture models with an infinite number of mixture components. A Dirichlet process (DP), parameterized by a base distribution $G_0$ and a concentration parameter $\alpha$, is used as a prior over the distribution $G$ of mixture components. For data points $X$, mixture component parameters $\theta$, and a parameterized distribution $F$, the DPMM can be written as [13]:

$$G|\alpha, G_0 \sim DP(\alpha, G_0)$$
$$\theta_i|G \sim G$$
$$x_i|\theta_i \sim F(\theta_i).$$

One type of DPMM can be implemented as an infinite Gaussian mixture model (IGMM) in which all parameters are inferred from the data [16]. Gibbs sampling is used to generate samples from the posterior distribution of the IGMM and adaptive rejection sampling [4] is used for the probabilities which are not in a standard form. After a "burn-in" period, unbiased samples from the posterior distribution of the IGMM can be drawn from the Gibbs sampler. A hard clustering can be found by drawing many such samples and using the sample with the highest joint likelihood of the class indicator variables. We use a modified IGMM implementation written by M. Mandel [1].

## 3 Latent Skill Discovery

To aid thinking about our algorithm, subgoals can be viewed as samples from the termination sets of *latent options* that are implicitly defined by the distribution of tasks, the chosen subgoal discovery algorithm, and the agent definition. Specifically, we define the latent options as those whose termination sets contain all of the sampled subgoal data and that maximize the expected discounted cumulative reward when used by a particular agent on a distribution of tasks (assuming optimal option policies given the termination sets). When many such maximizing sets exist, we assume that the latent options are one particular set from amongst these choices; for discussion, the particular choice does not matter, but it is important to have a single set.

Therefore, our goal is to recover the termination sets of the latent options from the sampled subgoal data; these can be used to construct a library of options that approximate the latent options and have the following desirable properties:

- Recall: The termination sets of the library options should contain a maximal portion of the termination sets of the latent options.
- Precision: The termination sets of the library options should contain minimal regions that are not in the termination sets of the latent options.
- Separability: The termination set of each library option should be entirely contained within the termination set of some single latent option.
- Minimality: A minimal number of options should be defined, while still meeting the above criteria. Ideally, this will be equal to the number of latent options.

Most of these properties are straightforward, but the importance of separability should be emphasized. Imagine an agent that faces a distribution of tasks with several latent options that need to be sequenced in various ways for each task. If a clustering breaks each latent option termination set into two options (minimality is violated, but separability is preserved), some exploration inefficiency

may be introduced, but each option will reliably terminate in a skill-appropriate state. However, if a clustering combines the termination sets of two latent options into that of a single library option, the library option becomes unreliable; when the functionality of a single latent option is needed, the combined option may exhibit behavior corresponding to either.

We cannot reason directly about latent options since we do not know what they are *a priori*, so we must estimate them with respect to the above constraints from sampled subgoal data alone. We assume that subgoal samples corresponding to the same latent option form a contiguous region on some manifold, which is reflected in the problem representation. If they do not, then our method cannot cluster and find skills; we view this as a failing of the representation and not of our methodology.

Under this assumption, clustering of sampled subgoals can be used to approximate latent option termination sets. We propose a method of converting clusters parameterized by Gaussians into termination sets that respect the recall and precision properties. Knowing the number of skills *a priori* or discovering the appropriate number of clusters from the data satisfies the minimality property. Separability is more complicated, but can be satisfied by any method that can handle overlapping clusters without merging them and that is not inherently biased toward a small number of skills. Methods like spectral clustering [14] that rely on point-wise distance metrics cannot easily handle cluster overlap and are unsuitable for this sort of task. In the general case where little is known about the number and nature of the latent options, IGMM-based clustering is an attractive choice, as it can model any number of clusters of arbitrary complexity; when clusters have a complex shape, an IGMM may over-segment the data, but this still produces separable options.

# 4 Algorithm

We present a general algorithm to discover latent options when using any particular subgoal discovery method and clustering algorithm. Note that some subgoal discovery methods discover state regions, rather than single states; in such cases, sampling techniques or a clustering algorithm such as NPClu [5] that can handle non-point data must be used. We then describe a specific implementation of the general algorithm that is used in our experiments.

## 4.1 General algorithm

Given an agent $\mathcal{A}$, task distribution $\tau$, subgoal discovery algorithm $\mathcal{D}$, and clustering algorithm $\mathcal{C}$:

1. Compute a set of sample agent-space subgoals $X = \{x_1, x_2, ..., x_n\}$, where $X = \mathcal{D}(\mathcal{A}, \tau)$.
2. Cluster the subgoals $X$ into clusters with parameters $\theta = \{\theta_1, \theta_2, ..., \theta_k\}$, where $\theta = \mathcal{C}(X)$. If the clustering method is parametric, then the elements of $\theta$ are cluster parameters, otherwise they are data point assignments to clusters.
3. Define option termination sets $\mathcal{T}_1, \mathcal{T}_2, ..., \mathcal{T}_k$, where $\mathcal{T}_i = \mathcal{M}(\theta_i)$, and $\mathcal{M}$ is a mapping from elements of $\theta$ to termination set definitions.
4. Instantiate and train options $\mathcal{O}_1, \mathcal{O}_2, ..., \mathcal{O}_k$ using $\mathcal{T}_1, \mathcal{T}_2, ..., \mathcal{T}_k$ as termination sets.

## 4.2 Experimental implementation

We now present an example implementation of the general algorithm that is used in our experiments. As to not confound error from our clustering method with error introduced by a subgoal discovery algorithm, we use a hand-coded binary salience function; the main contribution of this work is the clustering strategy that enables generalization and transfer, so we are not concerned with the details of any particular subgoal discovery algorithm. This also demonstrates the possible utility of our approach, even when automatic subgoal discovery is inappropriate or infeasible. More details on this are presented in the following sections.

First, a distribution of tasks and an RL agent are defined. We allow the agent to solve tasks drawn from this distribution while collecting subgoal state samples every time the salience function is triggered. This continues until 10,000 subgoal state samples are collected. These points are then clustered using one of two different clustering methods. Gaussian expectation-maximization (E-M),

for which we must provide the number of clusters *a priori*, provides an approximate upper bound on the performance of any clustering method based on a Gaussian mixture model. We compare this to IGMM-based clustering that must discover the number of clusters automatically. E-M is used as a baseline metric to separate error caused by not knowing the number of clusters *a priori* from error caused by using a Gaussian mixture model. Since E-M can get stuck in local minima, we run it 10 times and choose the clustering with the highest log-likelihood. For the IGMM-based clustering, we let the Gibbs sampler burn-in for 10,000 samples and then collect an additional 10,000 samples, from which we choose the sample with the highest joint likelihood of the class indicator variables as defined by Rasmussen [16].

We now must define a mapping function $\mathcal{M}$ that maps our clusters to termination sets. Both of our clustering methods return a list of $K$ sets of Gaussian means $\mu$ and covariances $\Sigma$. We would like to choose a ridge on each Gaussian to be the cluster's termination set boundary; thus, we use Mahalanobis distance from each cluster mean, where

$$D_i^{Mahalanobis}(x) = \sqrt{(x - \mu_i)^T \Sigma_i^{-1}(x - \mu_i)} \ ,$$

and the termination set $\mathcal{T}_i$ is defined as:

$$\mathcal{T}_i(x) = \left\{ \begin{array}{ll} 1 & : D_i^{Mahalanobis}(x) \leq \epsilon_i \\ 0 & : \text{otherwise,} \end{array} \right.$$

where $\epsilon_i$ is a threshold. An appropriate value for each $\epsilon_i$ is found automatically by calculating the maximum $D_i^{Mahalanobis}(x)$ of any of the subgoal state points $x$ assigned to the $i^{th}$ cluster. This makes each $\epsilon_i$ just large enough so that all the subgoal state data points assigned to the $i^{th}$ cluster are within the $\epsilon_i$-Mahalanobis distance of that cluster mean, satisfying both our recall and precision conditions. Note that some states can be contained in multiple termination sets.

Using these termination sets, we create options that are given to the agent for a 100 episode "gestation period", during which the agent can learn option policies using off-policy learning, but cannot invoke the options. After this period, the options can be invoked from any state.

## 5  Experiments

### 5.1  Light-Chain domain

We test the various implementations of our algorithm on a continuous domain similar to the Light-world domain [9], designed to provide intuition about the capabilities of our skill discovery method. In our version, the Light-Chain domain, an agent is placed in a $10 \times 10$ room that contains a primary beacon, a secondary beacon, and a goal beacon placed in random locations. If the agent moves within 1 unit of the primary beacon, the beacon becomes "activated" for 30 time steps. Similarly, if the agent moves within 1 unit of the secondary beacon while the primary beacon is activated, it also becomes activated for 30 time steps. The goal of the task is for the agent to move within 1 unit of the goal beacon while the secondary beacon is activated, upon which it receives a reward of 100, ending the episode. In all other states, the agent receives a reward of $-1$. Additionally, each beacon emits a uniquely colored light—either red, green, or blue—that is selected randomly for each task. Figure 1 shows two instances of the Light-Chain domain with different beacon locations and light color assignments.

There are four actions available to the agent in every state: move north, south, east, or west. The actions are stochastic, moving the agent between 0.9 and 1.1 units (uniformly distributed) in the specified direction. In the case of an action that would move an agent through a wall, the agent simply moves up to the wall and stops. The problem-space for this domain is 4-dimensional: The x-position of the agent, the y-position of the agent, and two boolean variables denoting whether or not the primary and secondary beacons are activated, respectively. The agent-space is 6-dimensional and defined by RGB range sensors that the agent is equipped with. Three of the sensors describe the north/south distance of the agent from each of the three colored lights (0 if the agent is at the light, positive values for being north of it, and negative vales for being south of it). The other three sensors are identical, but measure east/west distance. Since the beacon color associations change with every task, a portable top-level policy cannot be learned in agent space, but portable agent-space options can be learned that reliably direct the agent toward each of the lights.

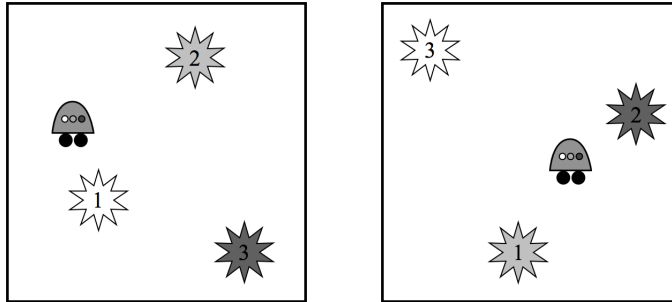

Figure 1: Two instances of the Light-Chain domain. The numbers 1–3 indicate the primary, secondary, and goal beacons respectively, while color signifies the light color each beacon emits. Notice that both beacon placement and color associations change between tasks.

The agent's salience function is defined as:

$$salient(t) = \left\{ \begin{array}{ll} 1 & : \text{At time } t, \text{ a beacon became activated for the first time in this episode.} \\ 0 & : \text{otherwise.} \end{array} \right.$$

Our algorithm clusters subgoal state data to create option termination conditions that generalize properly within a task and across tasks. In the Light-Chain domain, there are three latent options—one corresponding to each light color. Generalization within a task requires each option to terminate in any state within a 1 unit radius of its corresponding light color. However, if the agent only sees one task, all such states will be within some small fixed range of the other two lights; a termination set built from such data would not transfer to another task, since the relative positions of the lights would change. Thus, generalization across tasks requires each option to terminate when it is close to the proper light, regardless of the observed positions of the other two lights. When provided with data from many tasks, our algorithm can discover these relationships between agent-space variables and use them to define portable options. These options can then be used in each task, although in a different order for each, based on that task's color associations with the beacons.

Although we provide a broad subgoal (activate beacons) to the agent through the salience function, our algorithm does the work of discovering how many ways there are to accomplish these subgoals (three—one for each light color) and how to achieve each of these (get within 1 unit of that light). In each instance of the task, it is unknown which light color will correspond to each beacon. Therefore, it is not possible to define a skill that reliably guides the agent to a particular beacon (e.g. the primary beacon) and is portable across tasks. Instead, our algorithm discovers skills to navigate to particular lights, leading the agent to beacons by proxy. Note that this number of skills is independent of the number of beacons; if there were four possible colors of light, but only three beacons, four skills would be created so that the agent could perform well when presented with any three of the four colors in a given task. Similarly, such a setup can be used in other tasks where a broad subgoal is known, but the different means and number of ways of achieving it are unknown *a priori*.

## 5.2 Experimental structure

Two different agent types were used in our experiments: agents with and without options. The parameters for each agent type were optimized separately via a grid search. Top-level policies were learned using $\epsilon$-greedy SARSA($\lambda$) ($\alpha = 0.001$, $\gamma = 0.99$, $\lambda = 0.7$, $\epsilon = 0.1$ without options, $\alpha = 0.0005$, $\gamma = 0.99$, $\lambda = 0.9$, $\epsilon = 0.1$ with options) and the state-action value function was represented with a linear function approximator using the third order Fourier basis [8]. Option policies were learned off-policy (with an option reward of 1000 when in a terminating state), using Q($\lambda$) ($\alpha = 0.000025$, $\gamma = 0.99$, $\lambda = 0.9$) and the fifth order independent Fourier basis.

For the agents that discover options, we used the procedure outlined in the previous section to collect subgoal state samples and learn option policies. We compared these agents to an agent with perfect, hand-coded termination sets (each option terminated within 1 unit of a particular light) that followed the same learning procedure, but without the subgoal discovery step. After option policies were learned for 100 episodes, they were frozen and agent performance was measured for 10 episodes in

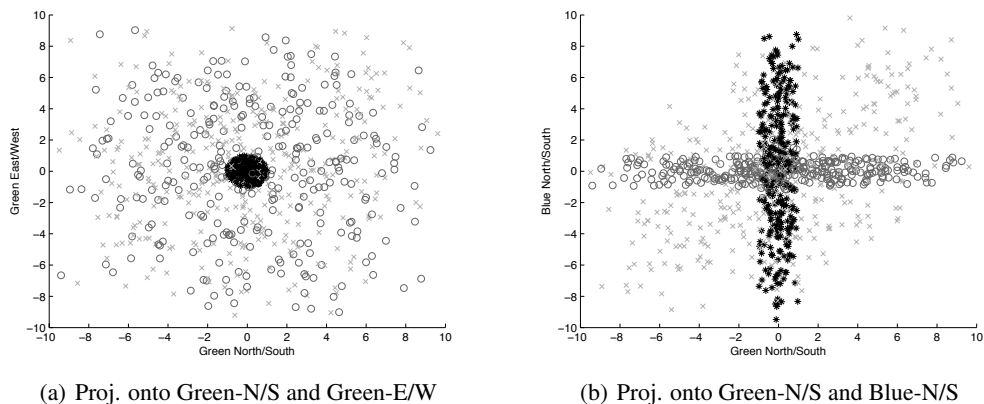

(a) Proj. onto Green-N/S and Green-E/W          (b) Proj. onto Green-N/S and Blue-N/S

Figure 2: IGMM clusterings of 6-dimensional subgoal data projected onto 2 dimensions at a time for visualization.

each of 1,000 novel tasks, with a maximum episode length of 5,000 steps and a maximum option execution time of 50 steps. After each task, the top-level policy was reset, but the option policies were kept constant. We compared performance of the agents using options to that of an agent without options, tested under the same conditions. This entire experiment was repeated 10 times.

## 6   Results

Figure 2 shows an IGMM-based clustering (only 1,000 points shown for readability), in which the original data points are projected onto 2 of the 6 agent-space dimensions at a time for visualization purposes, where cluster assignment is denoted with unique markers. It can be seen that three clusters (the intuitively optimal number) have been found. In 2(a), the data is projected onto the green north/south and green east/west dimensions. A central circular cluster is apparent, containing subgoals triggered by being near the green light. In 2(b), the north/south dimensions of two different light colors are compared. Here, there are two long clusters that each have a small variance with respect to one color and a large variance with respect to the other. These findings correspond to our intuitive notion of skills in this domain, in which an option should terminate when it is close to a particular light color, regardless of the positions of the other two lights. Note that these clusters actually overlap in 6 dimensions, not just in the projected view, since the activation radii of the beacons can occasionally overlap, depending on their placement.

Figure 3(a) compares the cumulative time it takes to solve 10 episodes for agents with no options, IGMM options, E-M options (with three clusters), and options with perfect, hand-coded termination sets. As expected, in all cases, options provide a significant learning advantage when facing a novel task. The agent using E-M options performs only slightly worse than the agent using perfect, hand-coded options, showing that clustering effectively discovers options in this domain and that very little error is introduced by using a Gaussian mixture model. Possibly more surprisingly, the agent using IGMM options performs equally as well as the agent using E-M options (making the lines difficult to distinguish in the graph), demonstrating that estimating the number of clusters automatically is feasible in this domain and introduces negligible error. In fact, the IGMM-based clustering finds three clusters in all 10 trials of the experiment.

Figure 3(b) shows the performance of agents using E-M options where the number of pre-specified clusters varies. As expected, the agent with three options (the intuitively optimal number of skills in this domain) performs the best, but the agents using five and six options still retain a significant advantage over an agent with no options. Most notably, when less than the optimal number of options are used, the agent actually performs worse than the baseline agent with no options. This confirms our intuition that option separability is more important than minimality. Thus, it seems that E-M may be effective if the designer can come up with a good approximation of the number of latent options, but it is critical to overestimate this number.

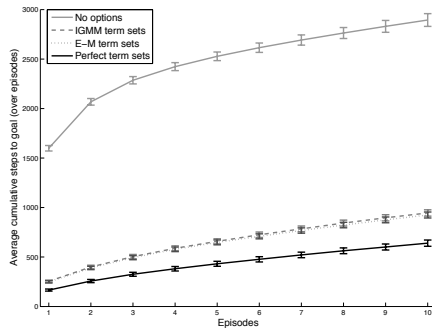
(a) Comparative performance of agents

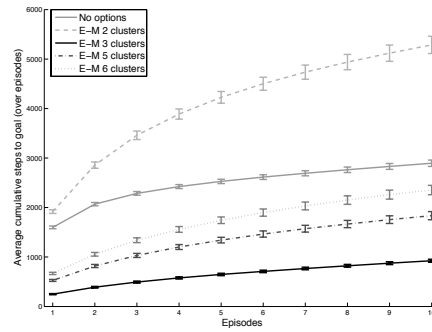
(b) E-M with varying numbers of clusters

Figure 3: Agent performance in Light-Chain domain with 95% confidence intervals

# 7 Discussion and Conclusions

We have demonstrated a general method for clustering agent-space subgoal data to form the termination sets of portable skills in the options framework. This method works in both discrete and continuous domains and can be used with any choice of subgoal discovery and clustering algorithms. Our analysis of the Light-Chain domain suggests that if the number of latent options is approximately known *a priori*, clustering algorithms like E-M can perform well. However, in the general case, IGMM-based clustering is able to discover an appropriate number of options automatically without sacrificing performance.

The collection and analysis of subgoal state samples can be computationally expensive, but this is a one-time cost. Our method is most relevant when a distribution of tasks is known ahead of time and we can spend computational time up front to improve agent performance on new tasks to be faced later, drawn from the same distribution. This can be beneficial when an agent will have to face a large number of related tasks, like in DRAM memory access scheduling [6], or for problems where fast learning and adaptation to non-stationarity is critical, such as automatic anesthesia administration [12].

In domains where traditional subgoal discovery algorithms fail or are too computationally expensive, it may be possible to define a salience function that specifies useful subgoals, while still allowing the clustering algorithm to decide how many skills are appropriate. For example, it is desirable to capture the queen in chess, but it may be beneficial to have several skills that result in different types of board configurations after taking the queen, rather than a single monolithic skill. Such a setup is advantageous when a broad subgoal is known *a priori*, but the various means and number of ways in which the subgoal might be accomplished are unknown, as in our Light-Chain experiment. This extends the possibility of skill discovery to a class of domains in which it may have previously been intractable.

An agent with a library of appropriate portable options ought to be able to learn novel tasks faster than an agent without options. However, as this library grows, the number of available actions actually increases and agent performance may begin to decline. This counter-intuitive notion, commonly known as the *utility problem*, reveals a fundamental problem with using skills outside the context of hierarchies. For skill discovery to be useful in larger problems, future work will have to address basic questions about how to automatically construct appropriate skill hierarchies that allow the agent to explore in simpler, more abstract action spaces as it gains more skills and competency.

**Acknowledgments**

We would like to thank Philip Thomas and George Konidaris for useful discussions. Scott Niekum and Andrew G. Barto were supported in part by the AFOSR under grant FA9550-08-1-0418.

## Footnotes

[1]Source code can be found at `http://mr-pc.org/work/`

# References

[1] Bram Bakker and Jürgen Schmidhuber. Hierarchical reinforcement learning based on subgoal discovery and subpolicy specialization. In *Proc. of the 8th Conference on Intelligent Autonomous Systems*, pages 438–445, 2004.

[2] A. G. Barto, S. Singh, and N. Chentanez. Intrinsically motivated learning of hierarchical collections of skills. In *Proc. of the International Conference on Developmental Learning*, pages 112–119, 2004.

[3] Bruce L. Digney. Learning hierarchical control structures for multiple tasks and changing environments. In *Proc. of the 5th Conference on the Simulation of Adaptive Behavior*. MIT Press, 1998.

[4] W. R. Gilks and P. Wild. Adaptive Rejection Sampling for Gibbs Sampling. *Journal of the Royal Statistical Society, Series C*, 41(2):337–348, 1992.

[5] M. Halkidi and M. Vazirgiannis. Npclu: An approach for clustering spatially extended objects. *Intell. Data Anal.*, 12:587–606, December 2008.

[6] Engin Ipek, Onur Mutlu, Jose F. Martinez, and Rich Caruana. Self-optimizing memory controllers: A reinforcement learning approach. *Computer Architecture, International Symposium on*, 0:39–50, 2008.

[7] Anders Jonsson and Andrew Barto. Causal graph based decomposition of factored mdps. *J. Mach. Learn. Res.*, 7:2259–2301, December 2006.

[8] G.D. Konidaris, S. Osentoski, and P.S. Thomas. Value function approximation in reinforcement learning using the fourier basis. In *Proceedings of the Twenty-Fifth Conference on Artificial Intelligence*, 2011.

[9] George Konidaris and Andrew G. Barto. Building portable options: Skill transfer in reinforcement learning. In *Proc. of the 20th International Joint Conference on Artificial Intelligence*, pages 895–900, 2007.

[10] George Konidaris and Andrew G. Barto. Skill discovery in continuous reinforcement learning domains using skill chaining. In *Advances in Neural Information Processing Systems 22*, pages 1015–1023, 2009.

[11] Amy McGovern and Andrew G. Barto. Automatic discovery of subgoals in reinforcement learning using diverse density. In *ICML*, pages 361–368, 2001.

[12] Brett Moore, Periklis Panousis, Vivek Kulkarni, Larry Pyeatt, and Anthony Doufas. Reinforcement learning for closed-loop propofol anesthesia: A human volunteer study. In *Innovative Applications of Artificial Intelligence*, 2010.

[13] R.M. Neal. Markov chain sampling methods for Dirichlet process mixture models. *Journal of computational and graphical statistics*, 9(2):249–265, 2000.

[14] Andrew Y. Ng, Michael I. Jordan, and Yair Weiss. On spectral clustering: Analysis and an algorithm. In *Advances in Neural Information Processing Systems*, pages 849–856. MIT Press, 2001.

[15] Marc Pickett and Andrew G. Barto. Policyblocks: An algorithm for creating useful macro-actions in reinforcement learning. In *ICML*, pages 506–513, 2002.

[16] Carl Edward Rasmussen. The infinite Gaussian mixture model. In *Advances in Neural Information Processing Systems 12*, pages 554–560. MIT Press, 2000.

[17] Özgür Şimşek and Andrew G. Barto. Using relative novelty to identify useful temporal abstractions in reinforcement learning. In *Proc. of the Twenty-First International Conference on Machine Learning*, pages 751–758, 2004.

[18] Özgür Şimşek and Andrew G. Barto. Skill characterization based on betweenness. In *NIPS*, pages 1497–1504, 2008.

[19] Richard Sutton, Doina Precup, and Satinder Singh. Between MDPs and semi-MDPs: A framework for temporal abstraction in reinforcement learning. *Artificial Intelligence*, 112:181–211, 1999.

[20] Richard S. Sutton and Andrew G. Barto. *Reinforcement Learning: An Introduction*. MIT Press, 1998.

[21] Sebastian Thrun and Anton Schwartz. Finding structure in reinforcement learning. In *Advances in Neural Information Processing Systems 7*, pages 385–392. MIT Press, 1995.

[22] Aaron Wilson, Alan Fern, Soumya Ray, and Prasad Tadepalli. Multi-task reinforcement learning: A hierarchical bayesian approach. In *In: ICML 07: Proceedings of the 24th international conference on Machine learning*, page 1015. ACM Press, 2007.

